# Recursive Estimation of Dynamic Modular RBF Networks

**Visakan Kadirkamanathan**
Automatic Control & Systems Eng. Dept.
University of Sheffield, Sheffield S1 4DU, UK
visakan@acse.sheffield.ac.uk

**Maha Kadirkamanathan**
Dragon Systems UK
Cheltenham GL52 4RW, UK
maha@dragon.co.uk

## Abstract

In this paper, recursive estimation algorithms for dynamic modular networks are developed. The models are based on Gaussian RBF networks and the gating network is considered in two stages: At first, it is simply a time-varying scalar and in the second, it is based on the state, as in the mixture of local experts scheme. The resulting algorithm uses Kalman filter estimation for the model estimation and the gating probability estimation. Both, 'hard' and 'soft' competition based estimation schemes are developed where in the former, the most probable network is adapted and in the latter all networks are adapted by appropriate weighting of the data.

## 1 INTRODUCTION

The problem of learning multiple modes in a complex nonlinear system is increasingly being studied by various researchers [2, 3, 4, 5, 6]. The use of a mixture of local experts [5, 6], and a conditional mixture density network [3] have been developed to model various modes of a system. The development has mainly been on model estimation from a given set of block data, with the model likelihood dependent on the input to the networks. A recursive algorithm for this static case is the approximate iterative procedure based on the block estimation schemes [6].

In this paper, we consider dynamic systems – developing a recursive algorithm is difficult since mode transitions have to be detected on-line whereas in the block scheme, search procedures allow optimal detection. Block estimation schemes for general architectures have been described in [2, 4]. However, unlike in those schemes, the algorithm developed here uses relationships based on Bayes law and Kalman filters and attempts to describe the dynamic system explicitly. The modelling is carried out by radial basis function (RBF) networks for their property that by preselecting the centres and widths, the problem can be reduced to a linear estimation.

## 2 DYNAMIC MODULAR RBF NETWORK

The dynamic modular RBF network consists of a number of models (or experts) to represent each nonlinear mode in a dynamical system. The models are based on the RBF networks with Gaussian function, where the RBF centre and width parameters are chosen *a priori* and the unknown parameters are only the linear coefficients $\mathbf{w}$. The functional form of the RBF network can be expressed as,

$$f(\mathbf{x}; \mathbf{p}) = \sum_{k=1}^{K} w_k g_k(\mathbf{x}) = \mathbf{w}^T \mathbf{g} \tag{1}$$

where $\mathbf{w} = [\ldots, w_k, \ldots]^T \in \Re^K$ is the linear weight vector and $\mathbf{g} = [\ldots, g_k(\mathbf{x}), \ldots]^T \in \Re_+^K$ are the radial basis functions, where,

$$g_k(\mathbf{x}) = \exp\left\{-0.5r^{-2} \|\mathbf{x} - \mathbf{m}_k\|^2\right\} \tag{2}$$

$\mathbf{m}_k \in \Re^M$ are the RBF centres or means and $r$ the width. The RBF networks are used for their property that having chosen appropriate RBF centre and width parameters $\mathbf{m}_k$, $r$, only the linear weights $\mathbf{w}$ need to be estimated for which fast, efficient and optimal algorithms exist.

Each model has an associated probability score of being the current underlying model for the given observation. In the first stage of the development, this probability is not determined from parametrised gating network as in the mixture of local experts [5] and the mixture density network [3], but is determined on-line as it varies with time. In dynamic systems, time information must be taken into account whereas the mixture of local experts use only the state information which is not sufficient in general, unless the states contain the necessary information. In the second stage, the probability is extended to represent both the time and state information explicitly using the expressions from the mixture of local experts. Recently, time and state information have been combined in developing models for dynamic systems such as the mixture of controllers [4] and the Input – Output HMM [2]. However, the scheme developed here is more explicit and is not as general as the above schemes and is recursive as opposed to block estimation.

## 3 RECURSIVE ESTIMATION

The problem of recursive estimation with RBF networks have been studied previously [7, 8] and the algorithms developed here is a continuation of that process. Let the set of input – output observations from which the model is to be estimated be,

$$\mathcal{Z}_N = \{z_n \mid n = 1, \ldots, N\} \tag{3}$$

where, $\mathcal{Z}_N$ includes all observations upto the $N$th data and $z_n$ is the $n$th data,

$$z_n = \{(\mathbf{x}_n, y_n) \mid \mathbf{x}_n \in \Re^M, y_n \in \Re\} \tag{4}$$

The underlying system generating the observations are assumed to be multi-modal (with known $H$ modes), with each observation satisfying the nonlinear relation,

$$y = f_h(\mathbf{x}) + \eta \tag{5}$$

where $\eta$ is the noise with unknown distribution and $f_h(\cdot) : \Re^M \mapsto \Re$ is the unknown underlying nonlinear function for the $h$th mode which generated the observation. Under assumptions of zero mean Gaussian noise and that the model can approximate the underlying function arbitrarily closely, the probability distribution,

$$p(z_n|\mathbf{w}^h, \mathcal{M}^n = \mathcal{M}_h, \mathcal{Z}_{n-1}) = (2\pi)^{-\frac{1}{2}} R_0^{-\frac{1}{2}} \exp\left\{-\frac{1}{2}R_0^{-1} \left|y_n - f_h(\mathbf{x}_n; \mathbf{w}^h)\right|^2\right\} \tag{6}$$

is Gaussian. This is the *likelihood* of the observation $z_n$ for the model $\mathcal{M}_h$, which in our case is the GRBF network, given model parameters $\mathbf{w}$ and that the $n$th observation was generated by $\mathcal{M}_h$. $R_0$ is the variance of the noise $\eta$. In general however, the model generating the $n$th observation is unknown and the likelihood of the $n$th observation is expanded to include $\gamma_n^h$ the *indicator variable*, as in [6],

$$p(z_n, \gamma_n | \mathbf{W}, \mathcal{M}, \mathcal{Z}_{n-1}) = \prod_{h=1}^{H} \left[ p(z_n | \mathbf{w}^h, \mathcal{M}^n = \mathcal{M}_h, \mathcal{Z}_{n-1}) p(\mathcal{M}^n = \mathcal{M}_h | \mathbf{x}_n, \mathcal{Z}^{n-1}) \right]^{\gamma_n^h} \tag{7}$$

Bayes law can be applied to the on-line or recursive parameter estimation,

$$p(\mathbf{W} | \mathcal{Z}_n, \mathcal{M}) = \frac{p(z_n | \mathbf{W}, \mathcal{M}, \mathcal{Z}_{n-1}) p(\mathbf{W} | \mathcal{Z}_{n-1}, \mathcal{M})}{p(z_n | \mathcal{Z}_{n-1}, \mathcal{M})} \tag{8}$$

and the above equation is applied recursively for $n = 1, \ldots, N$. The term $p(z_n | \mathcal{Z}_{n-1}, \mathcal{M})$ is the *evidence*. If the underlying system is unimodal, this will result in the optimal Kalman estimator and if we assign the *prior* probability distribution for the model parameters $p(\mathbf{w}^h | \mathcal{M}_h)$ to be Gaussian with mean $\mathbf{w}_0$ and covariance matrix (positive definite) $\mathbf{P}_0 \in \Re^{K \times K}$, which combines the likelihood and the prior to give the *posterior* probability distribution which at time $n$ is given by $p(\mathbf{w}^h | \mathcal{Z}_n, \mathcal{M}_h)$ which is also Gaussian,

$$p(\mathbf{w}^h | \mathcal{Z}_n, \mathcal{M}_h) = (2\pi)^{-\frac{K}{2}} \left| \mathbf{P}_n^h \right|^{-\frac{1}{2}} \exp \left\{ -\frac{1}{2} (\mathbf{w}^h - \mathbf{w}_n^h)^T \mathbf{P}_n^{h^{-1}} (\mathbf{w}^h - \mathbf{w}_n^h) \right\} \tag{9}$$

In the multimodal case also, the estimation for the individual model parameters decouple naturally with the only modification being that the likelihood used for the parameter estimation is now based on weighted data and given by,

$$p(z_n | \mathbf{w}^h, \mathcal{M}_h, \mathcal{Z}_{n-1}) = (2\pi)^{-\frac{1}{2}} (R_0 \gamma_n^{h^{-1}})^{-\frac{1}{2}} \exp \left\{ -\frac{1}{2} R_0^{-1} \gamma_n^h \left| y_n - f_h(\mathbf{x}_n ; \mathbf{w}^h) \right|^2 \right\} \tag{10}$$

The Bayes law relation (8) applies to each model. Hence, the only modification in the Kalman filter algorithm is that the noise variance for each model is set to $R_0 / \gamma_n^h$ and the resulting equations can be found in [7]. It increases the apparent uncertainty in the measurement output according to how likely the model is to be the true underlying mode, by increasing the noise variance term of the Kalman filter algorithm. Note that the term $p(\mathcal{M}^n = \mathcal{M}_h | \mathbf{x}_n, \mathcal{Z}^{n-1})$ is a time-varying scalar and does not influence the parameter estimation process.

The evidence term can also be determined directly from the Kalman filter,

$$p(z_n | \mathcal{M}_h, \mathcal{Z}_{n-1}) = (2\pi)^{-\frac{1}{2}} R_n^{h - \frac{1}{2}} \exp \left\{ -\frac{1}{2} R_n^{h^{-1}} \left| e_n^h \right|^2 \right\} \tag{11}$$

where the $e_n^h$ is the prediction error and $R_n^h$ is the innovation variance with,

$$e_n^h = y_n - \mathbf{w}_{n-1}^{h^T} \mathbf{g}_n \tag{12}$$

$$R_n^h = R_0 \gamma_n^{h^{-1}} + \mathbf{g}_n^T \mathbf{P}_{n-1}^h \mathbf{g}_n \tag{13}$$

This is also the likelihood of the $n^{th}$ observation given the model $\mathcal{M}$ and the past observations $\mathcal{Z}_{n-1}$. The above equation shows that the evidence term used in Bayesian model selection [9] is computed recursively, but for the specific priors $R_0$, $\mathbf{P}_0$. On-line Bayesian model selection can be carried out by choosing many different priors, effectively sampling the prior space, to determine the best model to fit the given data, as discussed in [7].

## 4   RECURSIVE MODEL SELECTION

Bayes law can be invoked to perform recursive or on-line model selection and this has been used in the derivation of the *multiple model algorithm* [1]. The multiple model algorithm has been used for the recursive identification of dynamical nonlinear systems [7]. Applying Bayes law gives the following relation:

$$p(\mathcal{M}_h|\mathcal{Z}_n) = \frac{p(z_n|\mathcal{M}_h, \mathcal{Z}_{n-1})p(\mathcal{M}_h|\mathcal{Z}_{n-1})}{p(z_n|\mathcal{Z}_{n-1})} \tag{14}$$

which can be computed recursively for $n = 1, \ldots, N$. $p(z_n|\mathcal{M}_h, \mathcal{Z}_{n-1})$ is the likelihood given in (11) and $p(\mathcal{M}_h|\mathcal{Z}_n)$ is the posterior probability of model $\mathcal{M}_h$ being the underlying model for the $n$th data given the observations $\mathcal{Z}_n$. The term $p(z_n|\mathcal{Z}_{n-1})$ is the normalising term given by,

$$p(z_n|\mathcal{Z}_{n-1}) = \sum_{h=1}^{H} p(z_n|\mathcal{M}_h, \mathcal{Z}_{n-1})p(\mathcal{M}_h|\mathcal{Z}_{n-1}) \tag{15}$$

The initial prior probabilities for models are assigned to be equal to $1/H$. The equations (11), (14) combined with the Kalman filter estimation equations is known as the multiple model algorithm [1].

Amongst all the networks that are attempting to identify the underlying system, the identified model is the one with the highest posterior probability $p(\mathcal{M}_h|\mathcal{Z}_n)$ at each time $n$, *ie.*,

$$\mathcal{M}^n = \arg\max_{\mathcal{M}_h} p(\mathcal{M}_h|\mathcal{Z}_n) \tag{16}$$

and hence can vary from time to time. This is preferred over the averaging of all the $H$ models as the likelihood is multimodal and hence modal estimates are sought. Predictions are based on this most probable model.

Since the system is dynamical, if the underlying model for the dynamics is known, it can be used to predict the estimates at the next time instant based on the current estimates, prior to observing the next data. Here, a first order Markov assumption is made for the mode transitions. Given that at the time instant $n - 1$ the given mode is $j$, it is predicted that the probability of the mode at time instant $n$ being $h$ is the transition probability $P_{hj}$. With $H$ modes, $\sum P_{hj} = 1$. The predicted probability of the mode being $h$ at time $n$ therefore is given by,

$$p_{n|n-1}(\mathcal{M}_h|\mathcal{Z}_{n-1}) = \sum_{j=1}^{H} P_{hj} p(\mathcal{M}_j|\mathcal{Z}_{n-1}) \tag{17}$$

This can be viewed as the prediction stage of the model selection algorithm. The predicted output of the system is obtained from the output of the model that has the highest predicted probability.

Given the observation $z_n$, the correction is achieved through the multiple model algorithm of (14) with the following modification:

$$p(\mathcal{M}_h|\mathcal{Z}_n) = \frac{p(z_n|\mathcal{M}_h, \mathcal{Z}_{n-1})p_{n|n-1}(\mathcal{M}_h|\mathcal{Z}_{n-1})}{p(z_n|\mathcal{Z}_{n-1})} \tag{18}$$

where modification to the prior has been made. Note that this probability is a time-varying scalar value and does not depend on the states.

## 5   HARD AND SOFT COMPETITION

The development of the estimation and model selection algorithms have thus far assumed that the indicator variable $\gamma_n^h$ is known. The $\gamma_n^h$ is unknown and an expected value must be used in the algorithm, which is given by,

$$\beta_n^h = \frac{p(z_n|\mathcal{M}^n = \mathcal{M}_h, \mathcal{Z}_{n-1})p_{n|n-1}(\mathcal{M}^n = \mathcal{M}_h|\mathcal{Z}_{n-1})}{p(z_n|\mathcal{Z}_{n-1})} \qquad (19)$$

Two possible methodologies can be used for choosing the values for $\gamma_n^h$. In the first scheme,

$$\gamma_n^h = 1 \quad \text{if} \quad \beta_n^h > \beta_n^j \text{ for all } j \neq h, \quad \text{and} \quad 0 \quad \text{otherwise} \qquad (20)$$

This results in *'hard' competition* where, only the model with the highest predicted probability undergoes adaptation using the Kalman filter algorithm while all other models are prevented from adapting. Alternatively, the expected value can be used in the algorithm,

$$\gamma_n^h = \beta_n^h \qquad (21)$$

which results in *'soft' competition* and all models are allowed to undergo adaptation with appropriate data weighting as outlined in section 3. This scheme is slightly different from that presented in [7]. Since the posterior probabilities of each mode effectively indicate which mode is dominant at each time $n$, changes can then be used as means of detecting mode transitions.

## 6   EXPERIMENTAL RESULTS

The problem chosen for the experiment is learning the inverse robot kinematics used in [3]. This is a two link rigid arm manipulator for which, given joint arm angles $(\theta_1, \theta_2)$, the end effector position in cartesian co-ordinates is given by,

$$\begin{aligned} x_1 &= L_1 \cos(\theta_1) - L_2 \cos(\theta_1 + \theta_2) \\ x_2 &= L_1 \sin(\theta_1) - L_2 \sin(\theta_1 + \theta_2) \end{aligned} \qquad (22)$$

$L_1 = 0.8$, $L_2 = 0.2$ being the arm lengths. The inverse kinematics learning problem requires the identification of the underlying mapping from $(x_1, x_2) \rightarrow (\theta_1, \theta_2)$, which is bi-modal. Since the algorithm is developed for the identification of dynamical systems, the data are generated with the joint angles being excited sinusoidally with differing frequencies within the intervals $[0.3, 1.2] \times [\pi/2, 3\pi/2]$. The first 1000 observations are used for training and the next 1000 observations are used for testing with the adaptation turned off. The models use 28 RBFs chosen with fixed parameters, the centres being uniformly placed on a $7 \times 4$ grid.

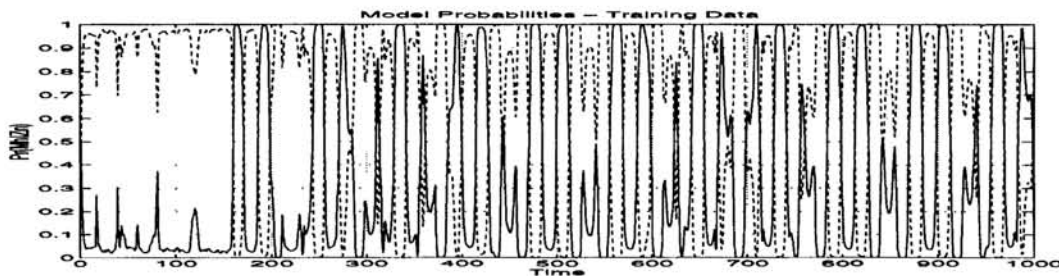

Figure 1: Learning inverse kinematics ('hard' competition): Model probabilities.

Figure 1 shows the model probabilities during training and shows the switching taking place between the two modes.

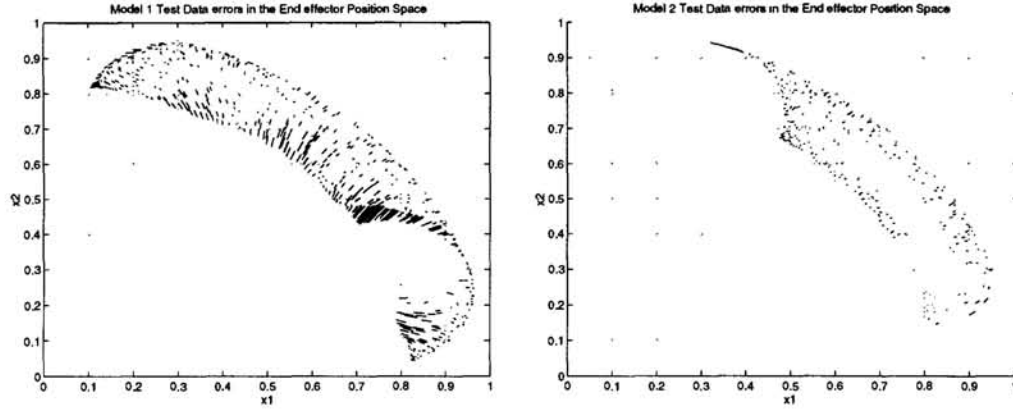

Figure 2: End effector position errors (test data) ('hard' competition): (a) Model 1 prediction (b) Model 2 prediction.

Figure 2 show the end effector position errors on the test data by both models 1 and 2 separately under the 'hard' competition scheme. The figure indicates the errors achieved by the best model used in the prediction – both models predicting in the centre of the input space where the function is multi-modal. This demonstrates the successful operation of the algorithm in the two RBF networks capturing some elements of the two underlying modes of the relationship. The best results on this learning task are: The RMSE on test data for this problem by the Mixture Density

Table 1: Learning Inverse Kinematics: Results

|  | Hard Competition | Soft Competition |
| --- | --- | --- |
| RMSE (Train) | 0.0213 | 0.0442 |
| RMSE (Test) | 0.0084 | 0.0212 |

Network is 0.0053 and by a single network is 0.0578 [3]. Note however that the algorithm here did not use state information and used only the time dependency.

## 7 PARAMETRISED GATING NETWORKS

The model parameters were determined explicitly based on the time information in the dynamical system. If the gating model probabilities are expressed as a function of the states, similar to [6],

$$p(\mathcal{M}_h|\mathbf{x}_n, \mathcal{Z}_{n-1}) = \exp\{\mathbf{a}^{h^T}\mathbf{g}\} \Big/ \sum_{h=1}^{H} \exp\{\mathbf{a}^{h^T}\mathbf{g}\} = \alpha_n^h \qquad (23)$$

where $\mathbf{a}^h$ are the gating network parameters. Note that the gating network shares the same basis functions as the expert models.

This extension to the gating networks does not affect the model parameter estimation procedure. The likelihood in (7) decomposes into a part for model parameter estimation involving output prediction error and a part for gating parameter estimation involving the indicator variable $\gamma_n$. The second part can be approximated to a Gaussian of the form,

$$p(\gamma_n|\mathbf{x}_n, \mathbf{a}^h, \mathcal{Z}_{n-1}) \approx (2\pi)^{-\frac{1}{2}} R_{g_0}^{h^{-\frac{1}{2}}} \exp\left\{-\frac{1}{2} R_{g_0}^{h^{-1}} |\gamma_n^h - \alpha_n^h|^2\right\} \qquad (24)$$

This approximation allows the extended Kalman filter algorithm to be used for gating network parameter estimation. The model selection equations of section 4 can be applied without any modification with the new gating probabilities. The choice of the indicator variable $\gamma_n^h$ can be based as before, resulting in either hard or soft competition. The necessary expressions in (21) are obtained through the Kalman filter estimates and the evidence values, for both the model and gating parameters. Note that this is different from the estimates used in [6] in the sense that, marginalisation over the model and gating parameters have been done here.

## 8  CONCLUSIONS

Recursive estimation algorithms for dynamic modular RBF networks have been developed. The models are based on Gaussian RBF networks and the gating is simply a time-varying scalar. The resulting algorithm uses Kalman filter estimation for the model parameters and the multiple model algorithm for the gating probability. Both, 'hard' and 'soft' competition based estimation schemes are developed where in the former, the most probable network is adapted and in the latter all networks are adapted by appropriate weighting of the data. Experimental results are given that demonstrate the capture of the switching in the dynamical system by the modular RBF networks. Extending the method to include the gating probability to be a function of the state are then outlined briefly. Work is currently in progress to experimentally demonstrate the operation of this extension.

## References

[1] Bar–Shalom, Y. and Fortmann, T. E. *Tracking and data association*, Academic Press, New York, 1988.

[2] Bengio, Y. and Frasconi, P. "An input output HMM architecture", In G. Tesauro, D. S. Touretzky and T. K. Leen (eds.) *Advances in Neural Information Processing Systems 7*, Morgan Kaufmann, CA: San Mateo, 1995.

[3] Bishop, C. M. "Mixture density networks", *Report NCRG/4288*, Computer Science Dept., Aston University, UK, 1994.

[4] Cacciatore, C. W. and Nowlan, S. J. "Mixtures of controllers for jump linear and nonlinear plants", In J. Cowan, G. Tesauro, and J. Alspector (eds.) *Advances in Neural Information Processing Systems 6*, Morgan Kaufmann, CA: San Mateo, 1994.

[5] Jacobs, R. A., Jordan, M. I., Nowlan, S. J. and Hinton, G. E. "Adaptive mixtures of local experts", *Neural Computation*, *3*: 79-87, 1991.

[6] Jordan, M. I. and Jacobs, R. A. "Hierarchical mixtures of experts and the EM algorithm", *Neural Computation*, *6*: 181-214, 1994.

[7] Kadirkamanathan, V. "Recursive nonlinear identification using multiple model algorithm", In *Proceedings of the IEEE Workshop on Neural Networks for Signal Processing V*, 171-180, 1995.

[8] Kadirkamanathan, V. "A statistical inference based growth criterion for the RBF network", In *Proceedings of the IEEE Workshop on Neural Networks for Signal Processing IV*, 12-21, 1994.

[9] MacKay, D. J. C. "Bayesian interpolation", *Neural Computation*, *4*: 415-447, 1992.